# Projection onto A Nonnegative Max-Heap

**Jun Liu**
Arizona State University
Tempe, AZ 85287, USA
j.liu@asu.edu

**Liang Sun**
Arizona State University
Tempe, AZ 85287, USA
sun.liang@asu.edu

**Jieping Ye**
Arizona State University
Tempe, AZ 85287, USA
jieping.ye@asu.edu

## Abstract

We consider the problem of computing the Euclidean projection of a vector of length $p$ onto a non-negative max-heap—an ordered tree where the values of the nodes are all nonnegative and the value of any parent node is no less than the value(s) of its child node(s). This Euclidean projection plays a building block role in the optimization problem with a non-negative max-heap constraint. Such a constraint is desirable when the features follow an ordered tree structure, that is, a given feature is selected for the given regression/classification task only if its parent node is selected. In this paper, we show that such Euclidean projection problem admits an analytical solution and we develop a top-down algorithm where the key operation is to find the so-called *maximal root-tree* of the subtree rooted at each node. A naive approach for finding the maximal root-tree is to enumerate all the possible root-trees, which, however, does not scale well. We reveal several important properties of the maximal root-tree, based on which we design a bottom-up algorithm with merge for efficiently finding the maximal root-tree. The proposed algorithm has a (worst-case) linear time complexity for a sequential list, and $O(p^2)$ for a general tree. We report simulation results showing the effectiveness of the max-heap for regression with an ordered tree structure. Empirical results show that the proposed algorithm has an expected linear time complexity for many special cases including a sequential list, a full binary tree, and a tree with depth 1.

## 1 Introduction

In many regression/classification problems, the features exhibit certain hierarchical or structural relationships, the usage of which can yield an interpretable model with improved regression/classification performance [25]. Recently, there have been increasing interests on structured sparisty with various approaches for incorporating structures; see [7, 8, 9, 17, 24, 25] and references therein. In this paper, we consider an ordered tree structure: a given feature is selected for the given regression/classification task only if its parent node is selected. To incorporate such ordered tree structure, we assume that the model parameter $\mathbf{x} \in \mathbb{R}^p$ follows the non-negative max-heap structure[1]:

$$P = \{\mathbf{x} \geq 0, x_i \geq x_j \ \forall (x_i, x_j) \in E^t\}, \tag{1}$$

where $T^t = (V^t, E^t)$ is a target tree with $V^t = \{x_1, x_2, \ldots, x_p\}$ containing all the nodes and $E^t$ all the edges. The constraint set $P$ implies that if $x_i$ is the parent node of a child node $x_j$ then the value of $x_i$ is no less than the value of $x_j$. In other words, if a parent node $x_i$ is 0, then any of its child nodes $x_j$ is also 0. Figure 1 illustrates three special tree structures: 1) a full binary tree, 2) a sequential list, and 3) a tree with depth 1.

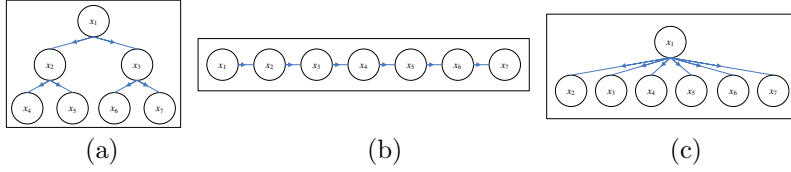

Figure 1: Illustration of a non-negative max-heap depicted in (1). Plots (a), (b), and (c) correspond to a full binary tree, a sequential list, and a tree with depth 1, respectively.

The set $P$ defined in (1) induces the so-called "heredity principle" [3, 6, 18, 24], which has been proven effective for high-dimensional variable selection. In a recent study [12], Li et al. conducted a meta-analysis of 113 data sets from published factorial experiments and concluded that an overwhelming majority of these real studies conform with the heredity principles. The ordered tree structure is a special case of the non-negative garrote discussed in [24] when the hierarchical relationship is depicted by a tree. Therefore, the asymptotic properties established in [24] are applicable to the ordered tree structrue. Several related approaches that can incorporate the ordered tree structure include the Wedge approach [17] and the hierarchical group Lasso [25]. The Wedge approach incorporates such ordering information by designing a penalty for the model parameter $\mathbf{x}$ as $\Omega(\mathbf{x}|P) = \inf_{\mathbf{t} \in P} \frac{1}{2} \sum_{i=1}^{p} (\frac{x_i^2}{t_i} + t_i)$, with tree being a sequential list. By imposing the mixed $\ell_1$-$\ell_2$ norm on each group formed by the nodes in the subtree of a parent node, the hierarchical group Lasso is able to incorporate such ordering information. The hierarchical group Lasso has been applied for multi-task learning in [11] with a tree structure, and the efficient computation was discussed in [10, 15]. Compared to Wedge and hierarchical group Lasso, the max-heap in (1) incorporates such ordering information in a direct way, and our simulation results show that the max-heap can achieve lower reconstruction error than both approaches.

In estimating the model parameter satisfying the ordered tree structure, one needs to solve the following constrained optimization problem:

$$\min_{\mathbf{x} \in P} f(\mathbf{x}) \tag{2}$$

for some convex function $f(\cdot)$. The problem (2) can be solved via many approaches including subgradient descent, cutting plane method, gradient descent, accelerated gradient descent, etc [19, 20]. In applying these approaches, a key building block is the so-called Euclidean projection of a vector $\mathbf{v}$ onto the convex set $P$:

$$\pi_P(\mathbf{v}) = \arg\min_{\mathbf{x} \in P} \frac{1}{2} \|\mathbf{x} - \mathbf{v}\|_2^2, \tag{3}$$

which ensures that the solution belongs to the constraint set $P$. For some special set $P$ (e.g., hyperplane, halfspace, and rectangle), the Euclidean projection admits a simple analytical solution, see [2]. In the literature, researchers have developed efficient Euclidean projection algorithms for the $\ell_1$-ball [5, 14], the $\ell_1/\ell_2$-ball [1], and the polyhedra [4, 22]. When $P$ is induced by a sequential list, a linear time algorithm was recently proposed in [26]. Without the non-negative constraints, problem (3) is the so-called isotonic regression problem [16, 21].

Our major technical contribution in this paper is the efficient computation of (3) for the set $P$ defined in (1). In Section 2, we show that the Euclidean projection admits an analytical solution, and we develop a top-down algorithm where the key operation is to find the so-called *maximal root-tree* of the subtree rooted at each node. In Section 3, we design a bottom-up algorithm with merge for efficiently finding the *maximal root-tree* by using its properties. We provide empirical results for the proposed algorithm in Section 4, and conclude this paper in Section 5.

## 2  Atda: A Top-Down Algorithm

In this section, we develop an algorithm in a top-down manner called Atda for solving (3). With the target tree $T^t = (V^t, E^t)$, we construct the input tree $T = (V, E)$ with the input vector $\mathbf{v}$, where $V = \{v_1, v_2, \ldots, v_p\}$ and $E = \{(v_i, v_j)|(x_i, x_j) \in E^t\}$. For the convenience of presenting our proposed algorithm, we begin with several definitions. We also provide some examples for elaborating the definitions in the supplementary file A.1.

**Definition 1.** *For a non-empty tree $T = (V, E)$, we define its root-tree as any non-empty tree $\tilde{T} = (\tilde{V}, \tilde{E})$ that satisfies: 1) $\tilde{V} \subseteq V$, 2) $\tilde{E} \subseteq E$, and 3) $\tilde{T}$ shares the same root as $T$.*

**Definition 2.** *For a non-empty tree $T = (V, E)$, we define $R(T)$ as the root-tree set containing all its root-trees.*

**Definition 3.** *For a non-empty tree $T = (V, E)$, we define*

$$m(T) = \max\left(\frac{\sum_{v_i \in V} v_i}{|V|}, 0\right), \tag{4}$$

*which equals the mean of all the nodes in $T$ if such mean is non-negative, and 0 otherwise.*

**Definition 4.** *For a non-empty tree $T = (V, E)$, we define its maximal root-tree as:*

$$M_{\max}(T) = \arg \max_{\tilde{T} = (\tilde{V}, \tilde{E}) : \tilde{T} \in R(T), m(\tilde{T}) = m_{\max}(T)} |\tilde{V}|, \tag{5}$$

*where*

$$m_{\max}(T) = \max_{\tilde{T} \in R(T)} m(\tilde{T}) \tag{6}$$

*is the maximal value of all the root-trees of the tree $T$. Note that if two root-trees share the same maximal value, (5) selects the one with the largest tree size.*

When $\tilde{T} = (\tilde{V}, \tilde{E})$ is a part of a "larger" tree $T = (V, E)$, i.e., $\tilde{V} \subseteq V$ and $\tilde{E} \subseteq E$, we can treat $\tilde{T}$ as a "super-node" of the tree $T$ with value $m(\tilde{T})$. Thus, we have the following definition of a super-tree (note that a super-tree provides a disjoint partition of the given tree):

**Definition 5.** *For a non-empty tree $T = (V, E)$, we define its super-tree as $S = (V_S, E_S)$, which satisfies: 1) each node in $V_S = \{T_1, T_2, \ldots, T_n\}$ is a non-empty tree with $T_i = (V_i, E_i)$, 2) $V_i \subseteq V$ and $E_i \subseteq E$, 3) $V_i \bigcap V_j = \emptyset, i \neq j$ and $V = \bigcup_{i=1}^{n} V_i$, and 4) $(T_i, T_j) \in E_S$ if and only if there exists a node in $T_j$ whose parent node is in $T_i$.*

## 2.1 Proposed Algorithm

We present the pseudo code for solving (3) in Algorithm 1. The key idea of the proposed algorithm is that, in the $i$-th call, we find $T_i = M_{\max}(T)$, the maximal root-tree of $T$, set $\tilde{\mathbf{x}}$ corresponding to the nodes of $T_i$ to $m_i = m_{\max}(T) = m(T_i)$, remove $T_i$ from the tree $T$, and apply Atda to the resulting trees one by one recursively.

---

**Algorithm 1** A Top-Down Algorithm: Atda

---

**Input:** the tree structure $T = (V, E)$, $i$
**Output:** $\tilde{\mathbf{x}} \in \mathbb{R}^p$
 1: Set $i = i + 1$
 2: Find the maximal root-tree of $T$, denoted by $T_i = (V_i, E_i)$, and set $m_i = m(T_i)$
 3: **if** $m_i > 0$ **then**
 4:　　Set $\tilde{x}_j = m_i, \forall v_j \in V_i$
 5:　　Remove the root-tree $T_i$ from $T$, denote the resulting trees as $\tilde{T}_1, \tilde{T}_2, \ldots, \tilde{T}_{r_i}$, and apply Atda($\tilde{T}_j$,$i$), $\forall j = 1, 2, \ldots, r_i$
 6: **else**
 7:　　Set $\tilde{x}_j = m_i, \forall v_j \in V_i$
 8: **end if**

---

## 2.2 Illustration & Justification

For a better illustration and justification of the proposed algorithm, we provide the analysis of Atda for a special case—the sequential list—in the supplementary file A.2.

Let us analyze Algorithm 1 for the general tree. Figure 2 illustrates solving (3) via Algorithm 1 for a tree with depth 3. Plot (a) shows a target tree $T^t$, and plot (b) denotes the input tree $T$. The dashed frame of plot (b) shows $M_{\max}(T)$, the maximal root-tree of $T$, and

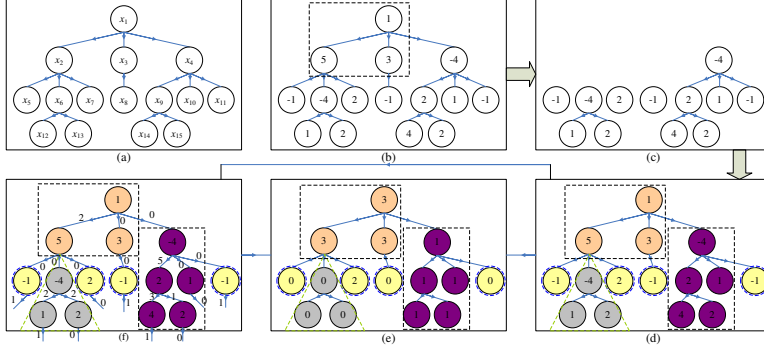

Figure 2: Illustration of Algorithm 1 for solving (3) for a tree with depth 3. Plot (a) shows the target tree $T^t$, and plots (b-e) illustrate Atda. Specifically, plot (b) denotes the input tree $T$, with the dashed frame displaying its maximal root-tree; plot (c) depicts the resulting trees after removing the maximal root-tree in plot (b); plot (d) shows the resulting super-tree (we treat each tree enclosed by the dashed frame as a super-node) of the algorithm; plot (e) gives the solution $\tilde{\mathbf{x}} \in \mathbb{R}^{15}$; and the edges of plot (f) show the dual variables, from which we can also obtain the optimal solution $\tilde{\mathbf{x}}$ (refer to the proof of Theorem 1).

we have $M_{\max}(T) = 3$. Thus, we set the corresponding entries of $\tilde{\mathbf{x}}$ to 3. Plot (c) depicts the resulting trees after removing the maximal root-tree in plot (b), and plot (d) shows the generated maximal root-trees (enclosed by dashed frame) by the algorithm. When treating each generated maximal root-tree as a super-node with the value defined in Definition 3, plot (d) is a super-tree of the input tree $T$. In addition, the super-tree is a max-heap, i.e., the value of the parent node is no less than the values of its child nodes. Plot (e) gives the solution $\tilde{\mathbf{x}} \in \mathbb{R}^{15}$. The edges of plot (f) correspond to the values of the dual variables, from which we can also obtain the optimal solution $\tilde{\mathbf{x}} \in \mathbb{R}^{15}$. Finally, we can observe that the non-zero entries of $\tilde{\mathbf{x}}$ constitute a cut of the original tree.

We verify the correctness of Algorithm 1 for the general tree in the following theorem. We make use of the KKT conditions and variational inequality [20] in the proof.

**Theorem 1.** $\tilde{\mathbf{x}} = \text{Atda}(T, 0)$ *provides the unique optimal solution to (3).*

**Proof:** As the objective function of (3) is strictly convex and the constraints are affine, it admits a unique solution. After running Algorithm 1, we obtain the sequences $\{T_i\}_{i=1}^k$ and $\{m_i\}_{i=1}^k$, where $k$ satisfies $1 \le k \le p$. It is easy to verify that the trees $T_i, i = 1, 2, \ldots, k$ constitute a disjoint partition of the input tree $T$. With the sequences $\{T_i\}_{i=1}^k$ and $\{m_i\}_{i=1}^k$, we can construct a super-tree of the input tree $T$ as follows: 1) we treat $T_i$ as a super-node with value $m_i$, and 2) we put an edge between $T_i$ and $T_j$ if there is an edge between the nodes of $T_i$ and $T_j$ in the input tree $T$. With Algorithm 1, we can verify that the resulting super-tree has the property that the value of the parent node is no less than its child nodes. Therefore, $\tilde{\mathbf{x}} = \text{Atda}(T, 0)$ satisfies $\tilde{\mathbf{x}} \in P$.

Let $\mathbf{x}^l$ and $\mathbf{v}^l$ denote a subset of $\mathbf{x}$ and $\mathbf{v}$ corresponding to the indices appearing in the subtree $T_l$, respectively. Denote $P^l = \{\mathbf{x}^l : \mathbf{x}^l \ge 0, x_i \ge x_j, (v_i, v_j) \in E_l\}$, $I_1 = \{l : m_l > 0\}$, $I_2 = \{l : m_l = 0\}$. Our proof is based on the following inequality:

$$\min_{\mathbf{x} \in P} \frac{1}{2}\|\mathbf{x} - \mathbf{v}\|_2^2 \ge \sum_{l \in I_1} \min_{\mathbf{x}^l \in P^l} \frac{1}{2}\|\mathbf{x}^l - \mathbf{v}^l\|_2^2 + \sum_{l \in I_2} \min_{\mathbf{x}^l \in P^l} \frac{1}{2}\|\mathbf{x}^l - \mathbf{v}^l\|_2^2, \qquad (7)$$

which holds as the left hand side has the additional inequality constraints compared to the right hand side. Our methodology is to show that $\tilde{\mathbf{x}} = \text{Atda}(T, 0)$ provides the optimal solution to the right hand side of (7), i.e.,

$$\tilde{\mathbf{x}}^l = \arg\min_{\mathbf{x}^l \in P^l} \frac{1}{2}\|\mathbf{x}^l - \mathbf{v}^l\|_2^2, \forall l \in I_1, \qquad (8)$$

$$\tilde{\mathbf{x}}^l = \arg\min_{\mathbf{x}^l \in P^l} \frac{1}{2}\|\mathbf{x}^l - \mathbf{v}^l\|_2^2, \forall l \in I_2, \qquad (9)$$

which, together with the fact $\frac{1}{2}\|\tilde{\mathbf{x}} - \mathbf{v}\|_2^2 \geq \min_{\mathbf{x} \in P} \frac{1}{2}\|\mathbf{x} - \mathbf{v}\|_2^2, \tilde{\mathbf{x}} \in P$ lead to our main argument. Next, we prove (8) by the KKT conditions, and prove (9) by the variational inequality [20].

Firstly, $\forall l \in I_1$, we introduce the dual variable $y_{ij}$ for the edge $(v_i, v_j) \in E_l$, and $y_{ii}$ if $v_i \in L_l$, where $L_l$ contains all the leaf nodes of the tree $T_l$. Denote the root of $T_l$ by $v_{r_l}$. For all $v_i \in V_l, v_i \neq v_{r_l}$, we denote its parent node by $v_{j_i}$, and for the root $v_{r_l}$, we denote $j_{r_l} = r_l$. We let

$$C_i^l = \{j | v_j \text{ is a child node of } v_i \text{ in the tree } T_l\}.$$

$$R_i^l = \{j | v_j \text{ is in the subtree of } T_l \text{ rooted at } v_i\}.$$

To prove (8), we verify that the primal variable $\tilde{\mathbf{x}} = \text{Atda}(T, 0)$ and the dual variable $\tilde{\mathbf{y}}$ satisfy the following KKT conditions:

$$\forall (v_i, v_j) \in E_l, \tilde{x}_i \geq \tilde{x}_j \quad \geq \quad 0 \tag{10}$$
$$\forall (v_i, v_j) \in E_l, (\tilde{x}_i - \tilde{x}_j)\tilde{y}_{ij} \quad = \quad 0 \tag{11}$$
$$\forall v_i \in L_l, \tilde{y}_{ii}\tilde{x}_i \quad = \quad 0 \tag{12}$$
$$\forall v_i \in V_l, \tilde{x}_i - v_i - \sum_{j \in C_i^l} \tilde{y}_{ij} + \tilde{y}_{j_i i} \quad = \quad 0 \tag{13}$$
$$\forall (v_i, v_j) \in E_l, \tilde{y}_{ij} \quad \geq \quad 0 \tag{14}$$
$$\forall v_i \in L_l, \tilde{y}_{ii} \quad \geq \quad 0, \tag{15}$$

where $\tilde{y}_{j_{r_l} r_l} = 0$ (Note that $\tilde{y}_{j_{r_l} r_l}$ is a dual variable, and it is introduced for the simplicity of presenting (12)), and the dual variable $\tilde{\mathbf{y}}$ is set as:

$$\tilde{y}_{ii} = 0, \forall i \in L_l, \tag{16}$$

$$\tilde{y}_{j_i i} = v_i - m_l + \sum_{j \in C_i^l} \tilde{y}_{ij}, \forall v_i \in V_l. \tag{17}$$

According to Algorithm 1, $\tilde{x}_i = m_l > 0, \forall v_i \in V_l, l \in I_1$. Thus, we have (10)-(12) and (15). It follows from (17) that (13) holds. According to (16) and (17), we have

$$\tilde{y}_{j_i i} = \sum_{j \in R_i^l} v_j - |R_i^l| m_l, \forall v_i \in V_l, \tag{18}$$

where $|R_i^l|$ denotes the number of elements in $R_i^l$, the subtree of $T_l$ rooted at $v_i$. From the nature of the maximal root-tree $T_l, l \in I_1$, we have $\sum_{j \in R_i^l} v_j \geq |R_i^l| m_l$. Otherwise, if $\sum_{j \in R_i^l} v_j < |R_i^l| m_l$, we can construct from $T_l$ a new root-tree $\bar{T}_l$ by removing the subtree of $T_l$ rooted at $v_i$, so that $\bar{T}_l$ achieves a larger value than $T_l$. This contradicts with the argument that $T_l, l \in I_1$ is the maximal root-tree of the working tree $T$. Therefore, it follows from (18) that (14) holds.

Secondly, we prove (9) by verifying the following optimality condition:

$$\langle \mathbf{x}^l - \tilde{\mathbf{x}}^l, \tilde{\mathbf{x}}^l - \mathbf{v}^l \rangle \geq 0, \forall \mathbf{x}^l \in P^l, l \in I_2, \tag{19}$$

which is the so-called variational inequality condition for $\tilde{\mathbf{x}}^l$ being the optimal solution to (9). According to Algorithm 1, if $l \in I_2$, we have $\tilde{x}_i = 0, \forall v_i \in V_l$. Thus, (19) is equivalent to

$$\langle \mathbf{x}^l, \mathbf{v}^l \rangle \leq 0, \forall \mathbf{x}^l \in P^l, l \in I_2. \tag{20}$$

For a given $\mathbf{x}^l \in P^l$, if $x_i = 0, \forall v_i \in V^l$, (20) naturally holds. Next, we consider $\mathbf{x}^l \neq 0$. Denote by $\bar{x}_1^l$ the minimal nonzero element in $\mathbf{x}^l$, and $T_l^1 = (V_l^1, E_l^1)$ a tree constructed by removing the nodes corresponding to the indices in the set $\{i : x_i^l = 0, v_i \in V_l\}$ from $T_l$. It is clear that $T_l^1$ shares the same root as $T_l$. It follows from Algorithm 1 that $\sum_{i:v_i \in V_l^1} v_i \leq 0$. Thus, we have

$$\langle \mathbf{x}^l, \mathbf{v}^l \rangle = \bar{x}_1^l \sum_{i:v_i \in V_l^1} v_i + \sum_{i:v_i \in V_l^1} (x_i - \bar{x}_1^l)v_i \leq \sum_{i:v_i \in V_l^1} (x_i - \bar{x}_1^l)v_i.$$

If $x_i^l = \bar{x}_1^l, \forall v_i \in V_l^1$, we arrive at (20). Otherwise, we set $r = 2$; denote by $\bar{x}_r^l$ the minimal nonzero element in the set $\{x_i - \sum_{j=1}^{r-1} \bar{x}_j^r : v_i \in V_l^{r-1}\}$, and $T_l^r = (V_l^r, E_l^r)$ a subtree of $T_l^{r-1}$ by removing those nodes with the indices in the set $\{i : x_i^l - \sum_{j=1}^{r-1} \bar{x}_j^l = 0, v_i \in V_l^{r-1}\}$. It is clear that $T_l^r$ shares the same root as $T_l^{r-1}$ and $T_l$ as well, so that it follows from Algorithm 1 that $\sum_{i:v_i \in V_l^r} v_i \leq 0$. Therefore, we have

$$\sum_{i:v_i \in V_l^{r-1}} (x_i - \sum_{j=1}^{r-1} \bar{x}_j^l)v_i = \bar{x}_r^l \sum_{i:v_i \in V_l^r} v_i + \sum_{i:v_i \in V_l^r} (x_i - \sum_{j=1}^{r} \bar{x}_j^l)v_i \leq \sum_{i:v_i \in V_l^r} (x_i - \sum_{j=1}^{r} \bar{x}_j^l)v_i. \quad (21)$$

Repeating the above process until $V_l^r$ is empty, we can verify that (20) holds. $\square$

For a better understanding of the proof, we make use of the edges of Figure 2 (f) to show the dual variables, where the edge connecting $v_i$ and $v_j$ corresponds to the dual variable $\tilde{y}_{ij}$, and the edge starting from the leaf node $v_i$ corresponds to the dual variable $\tilde{y}_{ii}$. With the dual variables, we can compute $\tilde{\mathbf{x}}$ via (13). We note that, for the maximal root-tree with a positive value, the associated dual variables are unique, but for the maximal root-tree with zero value, the associated dual variables may not be unique. For example, in Figure 2 (f), we set $\tilde{y}_{ii} = 1$ for $i = 12$, $\tilde{y}_{ii} = 0$ for $i = 13$, $\tilde{y}_{ij} = 2$ for $i = 6, j = 12$, and $\tilde{y}_{ij} = 2$ for $i = 6, j = 13$. It is easy to check that the dual variables can also be set as follows: $\tilde{y}_{ii} = 0$ for $i = 12$, $\tilde{y}_{ii} = 1$ for $i = 13$, $\tilde{y}_{ij} = 1$ for $i = 6, j = 12$, and $\tilde{y}_{ij} = 3$ for $i = 6, j = 13$.

## 3 Finding the Maximal Root-Tree

A key operation of Algorithm 1 is to find the maximal root-tree used in Step 2. A naive approach for finding the maximal root-tree of a tree $T$ is to enumerate all possible root-trees in the root-tree set $R(T)$, and identify the maximal root-tree via (5). We call such an approach Anae, which stands for a naive algorithm with enumeration. Although Anae is simple to describe, it has a very high time complexity (see the analysis given in supplementary file A.3). To this end, we develop Abuam (A Bottom-Up Algorithm with Merge). The underlying idea is to make use of the special structure of the maximal root-tree defined in (5) for avoiding the enumeration of all possible root-trees.

We begin the discussion with some key properties of the maximal root-tree, and the proof is given in the supplementary file A.4.

**Lemma 1.** *For a non-empty tree $T = (V, E)$, denote its maximal root-tree as $T_{\max} = (V_{\max}, E_{\max})$. Let $\tilde{T} = (\tilde{V}, \tilde{E})$ be a root-tree of $T_{\max}$. Assume that there are $n$ nodes $v_{i_1}, \ldots, v_{i_n}$, which satisfy: 1) $v_{i_j} \notin \tilde{V}$, 2) $v_{i_j} \in V$, and 3) the parent node of $v_{i_j}$ is in $\tilde{V}$. If $n \geq 1$, we denote the subtree of $T$ rooted at $v_{i_j}$ as $T^j = (V^j, E^j), j = 1, 2, \ldots, n$, $T_{\max}^j = (V_{\max}^j, E_{\max}^j)$ as the maximal root-trees of $T^j$, and $\tilde{m} = \max_{j=1,2,\ldots,n} m(T_{\max}^j)$. Then, the followings hold: (1) If $n = 0$, then $T_{\max} = \tilde{T} = T$; (2) If $n \geq 1$, $m(\tilde{T}) = 0$, and $\tilde{m} = 0$, then $T_{\max} = T$; (3) If $n \geq 1$, $m(\tilde{T}) > 0$, and $m(\tilde{T}) > \tilde{m}$, then $T_{\max} = \tilde{T}$; (4) If $n \geq 1$, $m(\tilde{T}) > 0$, and $m(\tilde{T}) \leq \tilde{m}$, then $V_{\max}^j \subseteq V_{\max}$, $E_{\max}^j \subseteq E_{\max}$ and $(v_{i_0}, v_{i_j}) \in E_{\max}$, $\forall j : m(T_{\max}^j) = \tilde{m}$; and (5) If $n \geq 1$, $m(\tilde{T}) = 0$, and $\tilde{m} > 0$, then $V_{\max}^j \subseteq V_{\max}$, $E_{\max}^j \subseteq E_{\max}$ and $(v_{i_0}, v_{i_j}) \in E_{\max}$, $\forall j : m(T_{\max}^j) = \tilde{m}$.*

For the convenience of presenting our proposed algorithm, we define the operation "merge" as follows:

**Definition 6.** *Let $T = (V, E)$ be a non-empty tree, and $T_1 = (V^1, E^1)$ and $T_2 = (V^2, E^2)$ be two trees that satisfy: 1) they are composed of a subset of the nodes and edges of $T$, i.e., $V^1 \in V$, $V^2 \in V$, $E^1 \in E$, and $E^2 \in E$; 2) they do not overlap, i.e., $V^1 \bigcap V^2 = \emptyset$, and $E^1 \bigcap E^2 = \emptyset$; and 3) in the tree $T$, $v_{i_2}$, the root node of $T_2$ is a child of $v_{i_1}$, a leaf node of $T_1$. We define the operation "merge" as $\tilde{T} = \mathrm{merge}(T_1, T_2, T)$, where $\tilde{T} = (\tilde{V}, \tilde{E})$ with $V = V_1 \bigcup V_2$ and $E = E_1 \bigcup E_2 \bigcup \{(v_{i_1}, v_{i_2})\}$.*

Next, we make use of Lemma 1 to efficiently compute the maximal root-tree, and present the pseudo code for Abuam in Algorithm 2. We provide the illustration of the proposed algorithm and the analysis of its computational cost in the supplementary file A.5 and A.6, respectively.

---

**Algorithm 2** A Bottom-Up Algorithm with Merge: Abuam

---

**Input:** the input tree $T = (V, E)$
**Output:** the maximal root-tree $T_{\max} = (V_{\max}, E_{\max})$
 1: Set $T_0 = (V_0, E_0)$, where $V_0 = \{x_{i_0}\}$ and $E_0 = \emptyset$
 2: **if** $v_{i_0}$ does not have a child node in $T$ **then**
 3:     Set $T_{\max} = T_0$, return
 4: **end if**
 5: **while** 1 **do**
 6:     Set $\tilde{m} = 0$, denote by $v_{i_1}, \ldots, v_{i_n}$ the $n$ nodes that satisfy: 1) $v_{i_j} \notin V_0$, 2) $v_{i_j} \in V$, and 3) the parent node of $v_{i_j}$ is in $V_0$, and denote by $T^j = (V^j, E^j), j = 1, 2, \ldots, n$ the subtree of $T$ rooted at $v_{i_j}$.
 7:     **if** $n = 0$ **then**
 8:         Set $T_{\max} = T_0 = T$, return
 9:     **end if**
10:     **for** $j = 1$ to $n$ **do**
11:         Set $T_{\max}^j = \mathrm{Abuam}(T^j)$, and $\tilde{m} = \max(m(T_{\max}^j), \tilde{m})$
12:     **end for**
13:     **if** $m(T_0) = \tilde{m} = 0$ **then**
14:         Set $T_{\max} = T$, return
15:     **else if** $m(\tilde{T}) > 0$ and $m(\tilde{T}) > \tilde{m}$ **then**
16:         Set $T_{\max} = T_0$, return
17:     **else**
18:         Set $T_0 = \mathrm{merge}(T_0, T_{\max}^j, T), \forall j : m(T_{\max}^j) = \tilde{m}$
19:     **end if**
20: **end while**

---

Making use of the fact that $T_0$ is always a valid root-tree of $T_{\max}$, the maximal root-tree of $T$, we can easily prove the following result using Lemma 1.

**Theorem 2.** $T_{\max}$ *returned by Algorithm 2 is the maximal root-tree of the input tree $T$.*

## 4 Numerical Simulations

**Effectiveness of the Max-Heap Structure** We test the effectiveness of the max-heap structure for linear regression $\mathbf{b} = A\mathbf{x}$, following the same experimental setting as in [17]. Specifically, the elements of $A \in \mathbb{R}^{n \times p}$ are generated i.i.d. from the Gaussian distribution with zero mean and standard derivation and the columns of $A$ are then normalized to have unit length. The regression vector $\mathbf{x}$ has $p = 127$ nonincreasing elements, where the first 10 elements are set as $x_i^* = 11 - i, i = 1, 2, \ldots, 10$ and the rest are zeros. We compared with the following three approaches: Lasso [23], Group Lasso [25], and Wedge [17]. Lasso makes no use of such ordering, while Wedge incorporates the structure by using an auxiliary ordered variable. For Group Lasso and Max-Heap, we try binary-tree grouping and list-tree grouping, where the associated trees are a full binary tree and a sequential list, respectively. The regression vector is put on the tree so that, the closer the node to the root, the larger the element is placed. In Group Lasso, the nodes appearing in the same subtree form a group. For the compared approaches, we use the implementations provided in [17][2]; and for Max-Heap, we solve (2) with $f(\mathbf{x}) = \frac{1}{2}\|A\mathbf{x} - \mathbf{b}\|_2^2 + \rho\|\mathbf{x}\|_1$ for some small $\rho = r \times \|A^T b\|_\infty$ (we set $r = 10^{-4}$, and $10^{-8}$ for the binary-tree grouping and list-tree grouping, respectively) and apply the accelerated gradient descent [19] approach with our proposed Euclidean projection. We compute the average model error $\|\mathbf{x} - \mathbf{x}^*\|^2$ over 50 independent runs, and report the results with a varying number of sample size $n$ in Figure 3 (a) & (b). As expected, GL-binary, MH-binary, Wedge, GL-list and MH-list outperform Lasso which does not incorporate such ordering information. MH-binary performs better than GL-binary, and MH-list performs better than Wedge and GL-list, due to the direct usage of such ordering information. In addition, the list-tree grouping performs better than the binary-tree grouping, as it makes better usage of such ordering information.

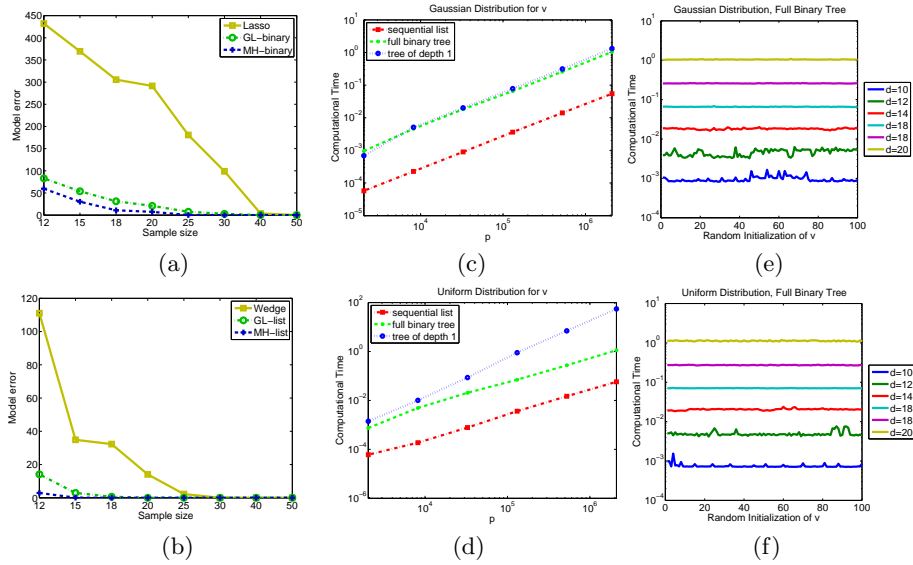

Figure 3: Simulation results. In plots (a) and (b), we show the average model error $\|\mathbf{x} - \mathbf{x}^*\|^2$ over 50 independet runs by different approaches with the full binary-tree ordering and the list-tree ordering. In plots (c) and (d), we report the computational time (in seconds) of the proposed Atda (averaged over 100 runs) with different randomly initialized input $\mathbf{v}$. In plots (e) and (f), we show the computational time of Atda over 100 runs.

**Efficiency of the Proposed Projection** We test the efficiency of the proposed Atda approach for solving the Euclidean projection onto the non-negative max-heap, equipped with our proposed Abuam approach for finding the maximal root-trees. In the experiments, we make use of the three tree structures as depicted in Figure 1, and try two different distributions: 1) Gaussian distribution with zero mean and standard derivation and 2) uniform distribution in $[0, 1]$ for randomly and independently generating the entries of the input $\mathbf{v} \in \mathbb{R}^p$. In Figure 3 (c) & (d), we report the average computational time (in seconds) over 100 runs under different values of $p = 2^{d+1} - 1$, where $d = 10, 12, \ldots, 20$. We can observe that, the proposed algorithm scales linearly with the size of $p$. In Figure 3 (e) & (f), we report the computational time of Atda over 100 runs when the ordered tree structure is a full binary tree. The results show that the computational time of the proposed algorithm is relatively stable for different runs, especially for larger $d$ or $p$. Note that, the source codes for our proposed algorithm have been included in the SLEP package [13].

## 5    Conclusion

In this paper, we have developed an efficient algorithm for the computation of the Euclidean projection onto a non-negative max-heap. The proposed algorithm has a (worst-case) linear time complexity for a sequential list, and $O(p^2)$ for a general tree. Empirical results show that: 1) the proposed approach deals with the ordering information better than existing approaches, and 2) the proposed algorithm has an expected linear time complexity for the sequential list, the full binary tree, and the tree of depth 1. It will be interesting to explore whether the proposed Abuam has a worst case linear (or linearithmic) time complexity for the binary tree. We plan to apply the proposed algorithms to real-world applications with an ordered tree structure. We also plan to extend our proposed approaches to the general hierarchical structure.

**Acknowledgments**

This work was supported by NSF IIS-0812551, IIS-0953662, MCB-1026710, CCF-1025177, NGA HM1582-08-1-0016, and NSFC 60905035, 61035003.

## Footnotes

[1]To deal with the negative model parameters, one can make use of the technique employed in [24], which solves the scaled version of the least square estimate.

[2]http://www.cs.ucl.ac.uk/staff/M.Pontil/software/sparsity.html

# References

[1] E. Berg, M. Schmidt, M. P. Friedlander, and K. Murphy. Group sparsity via linear-time projection. Tech. Rep. TR-2008-09, Department of Computer Science, University of British Columbia, Vancouver, July 2008.

[2] S. Boyd and L. Vandenberghe. *Convex Optimization*. Cambridge University Press, 2004.

[3] N. Choi, W. Li, and J. Zhu. Variable selection with the strong heredity constraint and its oracle property. *Journal of the American Statistical Association*, 105:354–364, 2010.

[4] Z. Dostál. Box constrained quadratic programming with proportioning and projections. *SIAM Journal on Optimization*, 7(3):871–887, 1997.

[5] J. Duchi, S. Shalev-Shwartz, Y. Singer, and C. Tushar. Efficient projection onto the $\ell_1$-ball for learning in high dimensions. In *International Conference on Machine Learning*, 2008.

[6] M. Hamada and C. Wu. Analysis of designed experiments with complex aliasing. *Journal of Quality Technology*, 24:130–137, 1992.

[7] J. Huang, T. Zhang, and D. Metaxas. Learning with structured sparsity. In *International Conference on Machine Learning*. 2009.

[8] L. Jacob, G. Obozinski, and J. Vert. Group lasso with overlap and graph lasso. In *International Conference on Machine Learning*, 2009.

[9] R. Jenatton, J.-Y. Audibert, and F. Bach. Structured variable selection with sparsity-inducing norms. Technical report, arXiv:0904.3523v2, 2009.

[10] R. Jenatton, J. Mairal, G. Obozinski, and F. Bach. Proximal methods for sparse hierarchical dictionary learning. In *International Conference on Machine Learning*, 2010.

[11] S. Kim and E. P. Xing. Tree-guided group lasso for multi-task regression with structured sparsity. In *International Conference on Machine Learning*, 2010.

[12] X. Li, N. Sundarsanam, and D. Frey. Regularities in data from factorial experiments. *Complexity*, 11:32–45, 2006.

[13] J. Liu, S. Ji, and J. Ye. *SLEP: Sparse Learning with Efficient Projections*. Arizona State University, 2009.

[14] J. Liu and J. Ye. Efficient Euclidean projections in linear time. In *International Conference on Machine Learning*, 2009.

[15] J. Liu and J. Ye. Moreau-yosida regularization for grouped tree structure learning. In *Advances in Neural Information Processing Systems*, 2010.

[16] R. Luss, S. Rosset, and M. Shahar. Decomposing isotonic regression for efficiently solving large problems. In *Advances in Neural Information Processing Systems*, 2010.

[17] C. Micchelli, J. Morales, and M. Pontil. A family of penalty functions for structured sparsity. In *Advances in Neural Information Processing Systems 23*, pages 1612–1623. 2010.

[18] J. Nelder. The selection of terms in response-surface models—how strong is the weak-heredity principle? *Annals of Applied Statistics*, 52:315–318, 1998.

[19] A. Nemirovski. *Efficient methods in convex programming*. Lecture Notes, 1994.

[20] Y. Nesterov. *Introductory Lectures on Convex Optimization: A Basic Course*. Kluwer Academic Publishers, 2004.

[21] P. M. Pardalos and G. Xue. Algorithms for a class of isotonic regression problems. *Algorithmica*, 23:211–222, 1999.

[22] S. Shalev-Shwartz and Y. Singer. Efficient learning of label ranking by soft projections onto polyhedra. *Journal of Machine Learning Research*, 7:1567–1599, 2006.

[23] R. Tibshirani. Regression shrinkage and selection via the lasso. *Journal of the Royal Statistical Society Series B*, 58(1):267–288, 1996.

[24] M. Yuan, V. R. Joseph, and H. Zou. Structured variable selection and estimation. *Annals of Applied Statistics*, 3:1738–1757, 2009.

[25] P. Zhao, G. Rocha, and B. Yu. The composite absolute penalties family for grouped and hierarchical variable selection. *Annals of Statistics*, 37(6A):3468–3497, 2009.

[26] L.W. Zhong and J.T. Kwok. Efficient sparse modeling with automatic feature grouping. In *International Conference on Machine Learning*, 2011.

